# Foundations for a Circuit Complexity Theory of Sensory Processing*

**Robert A. Legenstein & Wolfgang Maass**
Institute for Theoretical Computer Science
Technische Universität Graz, Austria
{legi, maass}@igi.tu-graz.ac.at

## Abstract

We introduce *total wire length* as salient complexity measure for an analysis of the circuit complexity of sensory processing in biological neural systems and neuromorphic engineering. This new complexity measure is applied to a set of basic computational problems that apparently need to be solved by circuits for translation- and scale-invariant sensory processing. We exhibit new circuit design strategies for these new benchmark functions that can be implemented within realistic complexity bounds, in particular with linear or almost linear total wire length.

## 1 Introduction

Circuit complexity theory is a classical area of theoretical computer science, that provides estimates for the complexity of circuits for computing specific benchmark functions, such as binary addition, multiplication and sorting (see, e.g. (Savage, 1998)). In recent years interest has grown in understanding the complexity of circuits for early sensory processing, both from the biological point of view and from the point of view of neuromorphic engineering (see (Mead, 1989)). However classical circuit complexity theory has provided little insight into these questions, both because its focus lies on a different set of computational problems, and because its traditional complexity measures are not tailored to those resources that are of primary interest in the analysis of neural circuits in biological organisms and neuromorphic engineering. This deficit is quite unfortunate since there is growing demand for energy-efficient hardware for sensory processing, and complexity issues become very important since the number $n$ of parallel inputs which such circuits have to handle is typically quite large (for example $n \geq 10^6$ in the case of many visual processing tasks).

We will follow traditional circuit complexity theory in assuming that the underlying graph of each circuit is a directed graph without cycles.[1] The most frequently considered complexity measures in traditional circuit complexity theory are the number (and types) of

*Research for this article was partially supported by the the Fonds zur Förderung der wissenschaftlichen Forschung (FWF), Austria, project P12153, and the NeuroCOLT project of the EC.

[1]Neural circuits in "wetware" as well as most circuits in analog VLSI contain in addition to feedforward connections also lateral and recurrent connections. This fact presents a serious obstacle for a direct mathematical analysis of such circuits. The standard mathematical approach is to model such circuits by larger feedforward circuits, where new "virtual gates" are introduced to represent the state of existing gates at later points in time.

gates, as well as the depth of a circuit. The latter is defined as the length of the longest directed path in the underlying graph, and is also interpreted as the computation time of the circuit. The focus lies in general on the classification of functions that can be computed by circuits whose number of gates can be bounded by a polynomial in the number $n$ of input variables. This implicitly also provides a polynomial – typically quite large – bound on the number of "wires" (defined as the edges in the underlying graph of the circuit).

We proceed on the assumption that the area (or volume in the case of neural circuits) occupied by wires is a severe bottleneck for physical implementations of circuits for sensory processing. Therefore we will not just count wires, but consider a complexity measure that provides an estimate for the total area or volume occupied by wires. In the cortex, neurons occupy an about 2 mm thick 3-dimensional sheet of "grey matter". There exists a strikingly general upper bound on the order of $10^5$ for the number of neurons under any $mm^2$ of cortical surface, and the total length of wires (axons and dendrites, including those running in the sheet of "white matter" that lies below the grey matter) under any $mm^2$ of cortical surface is estimated to be $\leq 8km = 8 \cdot 10^6 mm$ (Koch, 1999). Together this yields an upper bound of $\frac{8 \cdot 10^6}{10^5} n = 80 \cdot n$ mm for the wire length of the "average" cortical circuit involving $n$ neurons.

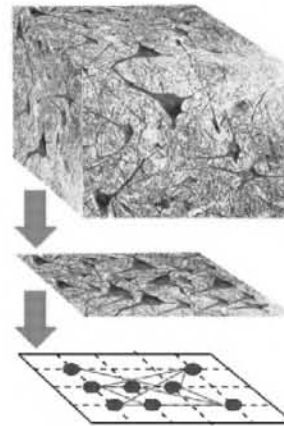

In order to arrive at a concise mathematical model we project each 3D cortical circuit into 2D, and assume for simplicity that its $n$ gates (neurons) occupy the nodes of a grid. Then for a circuit with $n$ gates, the total length of the horizontal components of all wires is on average $\leq 80 \cdot n$ mm $= 80 \cdot n$ $\cdot 10^{5/2} \cong 25300 \cdot n$ grid units. Here, one grid unit is the distance between adjacent nodes on the grid, which amounts to $10^{-5/2}$mm for an assumed density of $10^5$ neurons per $mm^2$ of cortical surface. Thus we arrive at a simple test for checking whether the total wire length of a proposed circuit design has a chance to be biologically realistic: *Check whether you can arrange its $n$ gates on the nodes of a grid in such a way that the total length of the horizontal components of all wires is $\leq 25300 \cdot n$ grid units.*

More abstractly, we define the following model:
*Gates, input- and output-ports of a circuit are placed on different nodes of a 2-dimensional grid (with unit distance 1 between adjacent nodes). These nodes can be connected by (unidirectional) wires that run through the plane in any way that the designer wants, in particular wires may cross and need not run rectilinearly (wires are thought of as running in the 3 dimensional space above the plane, without charge for vertical wire segments)[2]. We refer to the minimal value of the sum of all wire lengths that can be achieved by any such arrangement as the* total wire length *of the circuit.*

The attractiveness of this model lies in its mathematical simplicity, and in its generality. It provides a rough estimate for the cost of connectivity both in artificial (basically 2-dimensional) circuits and in neural circuits, where 2-dimensional wire crossing problems are apparently avoided (at least on a small scale) since dendritic and axonal branches are routed through 3-dimensional cortical tissue.

There exist quite reliable estimates for the order of magnitudes for the number $n$ of inputs, the number of neurons and the total wire length of biological neural circuits for sensory processing, see (Abeles, 1998; Koch, 1999; Shepherd, 1998; Braitenberg and Schüz, 1998).[3]

Collectively they suggest that only those circuit architectures for sensory processing are biologically realistic that employ a number of gates that is almost linear in the number $n$ of inputs, and a total wire length that is quadratic or subquadratic – with the additional requirement that the constant factor in front of the asymptotic complexity bound has a value close to 1. Since most asymptotic bounds in circuit complexity theory have constant factors in front that are much larger than 1, one really has to focus on circuit architectures with clearly subquadratic bounds for the total wire length. The complexity bounds for circuits that can realistically be implemented in VLSI are typically even more severe than for "wetware", and linear or almost linear bounds for the total wire length are desirable for that purpose.

In this article we begin the investigation of algorithms for basic pattern recognition tasks that can be implemented within this low-level complexity regime. The architecture of such circuits has to differ strongly from most previously proposed circuits for sensory processing, which usually involve at least 2 completely connected layers, since already complete connectivity between just two linear size 2-dimensional layers of a feedforward neural net requires a total wire length on the order of $n^{5/2}$. Furthermore a circuit which first selects a salient input segment consisting of a block of up to $m$ adjacent inputs in some 2-dimensional map, and then sends this block of $\leq m$ inputs in parallel to some central "pattern template matcher", typically requires a total wire length of $\Omega(n^{3/2} \cdot m)$ – even without taking the circuitry for the "selection" or the template matching into account.

## 2 Global Pattern Detection in 2-Dimensional Maps

For many important sensory processing tasks – such as for visual or somatosensory input – the input variables are arranged in a 2-dimensional map whose structure reflects spatial relationship in the outside world. We assume that local feature detectors are able to detect the presence of salient local features in their specific "receptive field", such as for example a center which emits

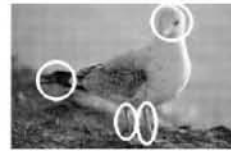

is estimated to be around $10^6$ (all estimates given are for primates, and they only reflect the order of magnitude). The total number of neurons that transmit sensory (mostly somatosensory) information to the cortex is estimated to be around $10^8$. In the subsequent sections we assume that these inputs represent the outputs of various local feature detectors for $n$ locations in some 2-dimensional map. Thus, if one assumes for example that on average there are 10 different feature detectors for each location on this map, one arrives at biologically realistic estimates for $n$ that lie between $10^5$ and $10^7$.

The total number of neurons in the primary visual cortex of primates is estimated to be around $10^9$, occupying an area of roughly $10^4$ mm$^2$ of cortical surface. There are up to $10^5$ neurons under one mm$^2$ of cortical surface, which yields a value of $10^{-5/2}$ mm for the distance between adjacent grid points in our model. The total length of axonal and dendritic branches below one mm$^2$ of cortical surface is estimated to be between 1 and 10 km, yielding up to $10^{11}$ mm total wire length for primary visual cortex. Thus if one assumes that 100 separate circuits are implemented in primary visual cortex, each of them can use $10^7$ neurons and a total wire length of $10^9$ mm. Hence realistic bounds for the complexity of a single one of these circuits for visual pattern recognition are $10^7 = n^{7/5}$ neurons (for $n = 10^5$), and a total wire length of at most $10^{11.5} = n^{2.3}$ grid units in the framework of our model.

The whole cortex receives sensory input from about $10^8$ neurons. It processes this input with about $10^{10}$ neurons and less than $10^{12}$ mm total wire length. If one assumes that $10^3$ separate circuits process this sensory information in parallel, each of them processing about 1/10th of the input (where again 10 different local feature detectors report about every location in a map), one arrives at $n = 10^6$ neurons for each circuit, and each circuit can use on average $n^{7/6}$ neurons and a total wire length of $10^{11.5} < n^2$ grid units in the sense of our model. The actual resources available for sensory processing are likely to be substantially smaller, since most cortical neurons and circuits are believed to have many other functions besides online sensory processing.

higher (or lower) intensity than its immediate surrounding, or a high-intensity line segment in a certain direction, the end of a line, a junction of line segments, or even more complex local visual patterns like an eye or a nose. The ultimate computational goal is to detect specific *global* spatial arrangements of such local patterns, such as the letter "T", or in the end also a human face, in a translation- and scale-invariant manner.

We formalize 2-dimensional global pattern detection problems by assuming that the input consists of arrays $\underline{a} = \langle a_1, \ldots, a_n \rangle, \underline{b} = \langle b_1, \ldots, b_n \rangle$, etc. of binary variables that are arranged on a 2-dimensional square grid[4]. Each index $i$ can be thought of as representing a location within some $\sqrt{n} \times \sqrt{n}$-square in the outside world. We assume that $a_i = 1$ if and only if feature $a$ is detected at location $i$ and that $b_i = 1$ if and only if feature $b$ is detected at location $i$. In our formal model we can reserve a subsquare within the 2-dimensional grid for each index $i$, where the input variables $a_i, b_i$, etc. are given on adjacent nodes of this grid[5]. Since we assume that this spatial arrangement of input variables reflects spatial relations in the outside world, many salient examples for global pattern detection problems require the computation of functions such as

$$P_D^n(\underline{a}, \underline{b}) = \begin{cases} 1, & \text{if there exist } i \text{ and } j \text{ so that } a_i = b_j = 1 \text{ and input location } j \\ & \text{is above and to the right of input location } i \\ 0, & \text{else} \end{cases}$$

**Theorem 2.1** *The function $P_D^n$ can be computed – and witnesses $i$ and $j$ with $a_i = b_j = 1$ can be exhibited if they exist – by a circuit with total wire length $O(n)$, consisting of $O(n)$ Boolean gates of fan-in 2 (and fan-out 2) in depth $O(\log n \cdot \log \log n)$.*
*The depth of the circuit can be reduced to $O(\log n)$ if one employs threshold gates[6] with fan-in $\log n$. This can also be done with total wire length $O(n)$.*

*Proof (sketch)* At first sight it seems that $P_D^n$ needs complete connectivity on the plane because of its global character. However, we show that there exists a divide and conquer approach with rather small communication cost.

Divide the input plane into four sub-squares $C_1, \ldots, C_4$ (see Figure 1a). We write $\underline{a}^1, \ldots, \underline{a}^4$ and $\underline{b}^1, \ldots, \underline{b}^4$ for the restrictions of the input to these four sub-areas and assume that the following values have already been computed for each sub-square $C_i$:

- The x-coordinate of the leftmost occurrence of feature $a$ in $C_i$
- The x-coordinate of the rightmost occurrence of feature $b$ in $C_i$
- The y-coordinate of the lowest occurrence of feature $a$ in $C_i$
- The y-coordinate of the highest occurrence of feature $b$ in $C_i$
- The value of $P_D^{n/4}(\underline{a}^i, \underline{b}^i)$

We employ a merging algorithm that uses this information to compute corresponding values for the whole input plane. The first four values can be computed by comparison-like

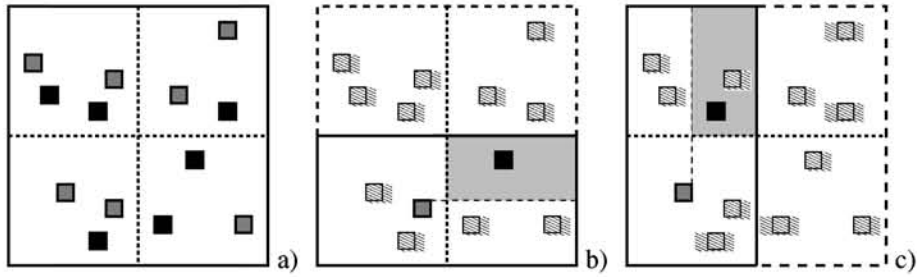

Figure 1: The 2-dimensional input plane. Occurrences of features in $\underline{a}$ are indicated by light squares, and occurrences of features in $\underline{b}$ are indicated by dark squares. Divide the input area into four sub-squares (a). Merging horizontally adjacent sub-squares (b). Merging vertically adjacent sub-squares (c).

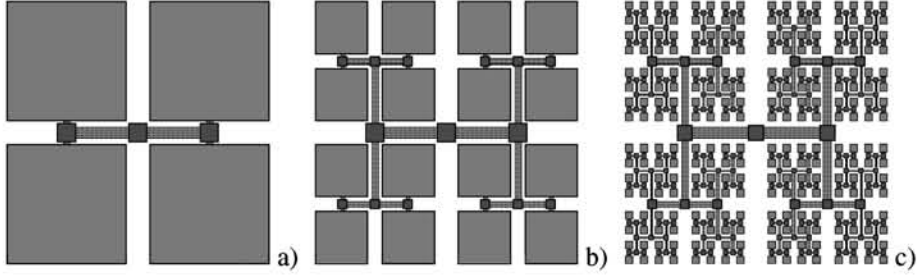

Figure 2: The H-tree construction. Black squares represent sub-circuits for the merging algorithm. The shaded areas contain the leaves of the tree. The lightly striped areas represent busses of wires that run along the edges of the H-Tree. The H-tree $H_1$ divides the input-area into four sub-squares (a). To construct $H_2$, replace the leaves of $H_1$ by H-trees $H_1$ (b). To construct $H_k$, replace the leaves of $H_1$ by H-trees $H_{k-1}$ (c).

operations. The computation of $P_D^n(\underline{a}, \underline{b})$ can be sketched as follows: First, check whether $P_D^{n/4}(\underline{a}^i, \underline{b}^i) = 1$ for some $i \in \{1, \ldots, 4\}$. Then, check the spatial relationships between feature occurrences in adjacent sub-squares. When checking spatial relationships between features from two horizontally adjacent sub-squares, only the lowest and the highest feature occurrence is crucial for the value of $P_D^n$ (see Figure 1b). This is true, since the x-coordinates are already separated. When checking spatial relationships of features from two vertically adjacent sub-squares, only the leftmost and the rightmost feature occurrence is crucial for the value of $P_D^n$ (see Figure 1c). This is true, since the y-coordinates are already separated. When checking spatial relationships of features from the lower left and the upper right sub-squares, it suffices to check whether there is an a-feature occurrence in the lower left and a b-feature occurrence in the upper right sub-square. Hence, one can reduce the amount of information needed from each sub-square to $O(\log n/4)$ bits.

In the remaining part of the proof sketch, we present an efficient layout for a circuit that implements this recursive algorithm. We need a layout strategy that is compatible with the recursive two-dimensional division of the input plane. We adopt for this purpose a well known design strategy: the H-tree (see (Mead and Rem, 1979)). An H-tree is a recursive tree-layout on the 2-dimensional plane. Let $H_k$ denote such a tree with $4^k$ leaves. The layout of $H_1$ is illustrated in Figure 2a. To construct an H-Tree $H_k$, build an H-tree $H_1$ and replace its four leaves by H-trees $H_{k-1}$ (see Figure 2b,c).

We need to modify the H-tree construction of Mead and Rum to make it applicable to

our problem. The inner nodes of the tree are replaced by sub-circuits that implement the merging algorithm. Furthermore, each edge of the H-tree is replaced by a "bus" consisting of $O(\log m)$ wires if it originates in an area with $m$ inputs. It is not difficult to show that this layout uses only linear total wire length. ∎

The linear total wire length of this circuit is up to a constant factor *optimal* for any circuit whose output depends on all of its $n$ inputs. Note that most connections in this circuit are local, just like in a biological neural circuit. Thus, we see that minimizing total wire length tends to generate biology-like circuit structures.

The next theorem shows that one can compute $P_D^n$ faster (i.e. by a circuit with smaller depth) if one can afford a somewhat larger total wire length. This circuit construction, that is based on AND/OR gates of limited fan-in $\Delta$, has the additional advantage that it can not just exhibit *some* pair $\langle i, j \rangle$ as witness for $P_D^n(\underline{a}, \underline{b}) = 1$ (provided such witness exists), but it can exhibit in addition *all* $j$ that can be used as witness together with some $i$. This property allows us to "chain" the global pattern detection problem formalized through the function $P_D^n$, and to decide within the same complexity bound whether for any fixed number $k$ of input vectors $\underline{a}^{(1)}, \ldots, \underline{a}^{(k)}$ from $\{0,1\}^n$ there exist locations $i^{(1)}, \ldots, i^{(k)}$ so that $a_{i^{(m)}}^{(m)} = 1$ for $m = 1, \ldots, k$ and location $i^{(m+1)}$ lies to the right and above location $i^{(m)}$ for $m = 1, \ldots, k-1$. In fact, one can also compute a $k$-tuple of witnesses $i^{(1)}, \ldots, i^{(k)}$ within the same complexity bounds, provided it exists. This circuit design is based on an efficient layout for prefix computations.

**Theorem 2.2** *For any given $n$ and $\Delta \in \{2, \ldots, \sqrt{n}\}$ one can compute the function $P_D^n$ in depth $O(\frac{\log n}{\log \Delta})$ by a feed-forward circuit consisting of $O(n)$ AND/OR gates of fan-in $\leq \Delta$, with total wire length $O(n \cdot \Delta \cdot \frac{\log n}{\log \Delta})$.* ∎

Another essential ingredient of translation- and scale-invariant global pattern recognition is the capability to detect whether a local feature $c$ occurs in the middle between locations $i$ and $j$ where the local features $a$ and $b$ occur. This global pattern detection problem is formalized through the following function $P_I^n : \{0,1\}^{3n} \rightarrow \{0,1\}$:

*If $\sum \underline{a} = \sum \underline{b} = 1$ then $P_I^n(\underline{a}, \underline{b}, \underline{c}) = 1$, if and only if there exist $i, j, k$ so that input location $k$ lies on the middle of the line between locations $i$ and $j$, and $a_i = b_j = c_k = 1$.*

This function $P_I^n$ can be computed very fast by circuits with the least possible total wire length (up to a constant factor), using threshold gates of fan-in up to $\sqrt{n}$:

**Theorem 2.3** *The function $P_I^n$ can be computed – and witnesses can be exhibited – by a circuit with total wire length and area $O(n)$, consisting of $O(n)$ Boolean gates of fan-in 2 and $O(\sqrt{n})$ threshold gates of fan-in $\sqrt{n}$ in depth 7.*

The design of the circuit exploits that the computation of $P_I^n$ can be reduced to the solution of two closely related 1-dimensional problems. ∎

## 3 Discussion

There exists a very large literature on neural circuits for translation-invariant pattern recognition see http://www.cnl.salk.edu/~wiskott/Bibliographies/Invariances.html. Unfortunately there exists substantial disagreement regarding the interpretation of existing approaches see http://www.ph.tn.tudelft.nl/PRInfo/shift/maillist.html. Virtually all positive results are based on computer simulations of small circuits, or on learning algorithms for concrete neural networks with a fixed input size $n$ on the order of 20 or 30, without an analysis how the required number of gates and the area or volume occupied by wires scale

up with the input size. The computational performance of these networks is often reported in an anecdotical manner.

The goal of this article is to show that circuit complexity theory may become a useful ingredient for understanding the computational strategies of biological neural circuits, and for extracting from them portable principles that can be applied to novel artificial circuits[7]. For that purpose we have introduced the total wire length as an abstract complexity measure that appears to be among the most salient ones in this context, and which can in principle be applied both to neural circuits in the cortex and to artificial circuitry. We would like to argue that only those computational strategies that can be implemented with subquadratic total wire length have a chance to reflect aspects of cortical information processing, and only those with almost linear total wire length are implementable in special purpose VLSI-chips for real-world sensory processing tasks.[8] The relevance of the total wire length of cortical circuits has been emphasized by numerous neuroscientists, from Cajal (see for example p. 14 in (Cajal, 1995)) to (Chklovskii and Stevens, 2000). On the other hand the total wire length of a circuit layout is also closely related to the area required by a VLSI implementation of such a circuit (see (Savage, 1998)).

We have formalized some basic computational problems, that appear to underly various translation- and scale-invariant sensory processing tasks, as a first set of benchmark functions for a circuit complexity theory of sensory processing. We have presented designs for circuits that compute these benchmark functions with small – in most cases linear or almost linear – total wire length (and constant factors of moderate size). The computational strategies of these circuits differ strongly from those that have been considered in previous approaches, which failed to take the limitations imposed by the realistically available amount of total wire length into account.

## Footnotes

[2]We will allow that a wire from a gate may branch and provide input to several other gates. For reasonable bounds on the maximal fan-out ($10^4$ in the case of neural circuits) this is realistic both for neural circuits and for VLSI.

[3]The number of neurons that transmit information from the retina (via the thalamus) to the cortex

[4]Whenever needed we assume for simplicity that $n$ is such that $\sqrt{n}, \log n$ etc. are natural numbers. The arrangement of the input variables an the grid will in general leave many nodes empty, which can be occupied by gates of the circuit.

[5]To make this more formal one can assume that indices $i$ and $j$ represent pairs $\langle i_1, i_2 \rangle, \langle j_1, j_2 \rangle$ of coordinates. Then "input location $j$ is above and to the right of input location $i$" means: $i_1 < j_1$ and $i_2 < j_2$. The circuit complexity of variations of the function $P_D^n$ where one or both of the "<" are replaced by "$\leq$" is the same.

[6]A threshold gate computes a Boolean function $T : \{0,1\}^k \to \{0,1\}$ of the form $T(x_1, \ldots, x_k) = 1 \Leftrightarrow \sum_{i=1}^k w_i x_i \geq w_0$.

[7]We do *not* want to argue that learning plays no role in the design and optimization of circuits for specific sensory processing tasks; on the contrary. But one of the few points where the discussion from http://www.ph.tn.tudelft.nl/PRInfo/shift/maillist.html agreed is that translation- and scale-invariant pattern recognition is a task which is so demanding, that learning algorithms have to be supported by pre-existing circuit structures.

[8]Of course there are other important complexity measures for circuits – such as energy consumption – besides those that have been addressed in this article.

# References

Abeles, M. (1998). *Corticonics: Neural Circuits of the Cerebral Cortex*, Cambridge Univ. Press.

Braitenberg, V., Schüz, A. (1998). *Cortex: Statistics and Geometry of Neuronal Connectivity*, 2nd ed., Springer Verlag.

Cajal, S.R. (1995). *Histology of the Nervous System*, volumes 1 and 2, Oxford University Press (New York).

Chklovskii, D.B. and Stevens, C.F. (2000). Wiring optimization in the brain. *Advances in Neural Information Processing Systems* vol. 12, MIT Press, 103-107.

Koch, C. (1999). *Biophysics of Computation*, Oxford Univ. Press.

Lazzaro, J., Ryckebusch, S., Mahowald, M. A., Mead, C. A. (1989). Winner-take-all networks of $O(n)$ complexity. *Advances in Neural Information Processing Systems*, vol. 1, Morgan Kaufmann (San Mateo), 703-711.

Mead, C. and Rem M. (1979). Cost and performance of VLSI computing structures. *IEEE J. Solid State Circuits* SC-14(1979), 455-462.

Mead, C. (1989). *Analog VLSI and Neural Systems*. Addison-Wesley (Reading, MA, USA).

Savage, J. E. (1998). *Models of Computation: Exploring the Power of Computing*. Addison-Wesley (Reading, MA, USA).

Shepherd, G. M. (1998). *The Synaptic Organization of the Brain*, 2nd ed., Oxford Univ. Press.

